# Multiparty Differential Privacy via Aggregation of Locally Trained Classifiers

**Manas A. Pathak**
Carnegie Mellon University
Pittsburgh, PA
manasp@cs.cmu.edu

**Shantanu Rane**
Mitsubishi Electric Research Labs
Cambridge, MA
rane@merl.com

**Bhiksha Raj**
Carnegie Mellon University
Pittsburgh, PA
bhiksha@cs.cmu.edu

## Abstract

As increasing amounts of sensitive personal information finds its way into data repositories, it is important to develop analysis mechanisms that can derive aggregate information from these repositories without revealing information about individual data instances. Though the differential privacy model provides a framework to analyze such mechanisms for databases belonging to a single party, this framework has not yet been considered in a multi-party setting. In this paper, we propose a privacy-preserving protocol for composing a differentially private aggregate classifier using classifiers trained locally by separate mutually untrusting parties. The protocol allows these parties to interact with an untrusted curator to construct additive shares of a perturbed aggregate classifier. We also present a detailed theoretical analysis containing a proof of differential privacy of the perturbed aggregate classifier and a bound on the excess risk introduced by the perturbation. We verify the bound with an experimental evaluation on a real dataset.

## 1 Introduction

In recent years, individuals and corporate entities have gathered large quantities of personal data. Often, they may wish to contribute the data towards the computation of functions such as various statistics, responses to queries, classifiers etc. In the process, however, they risk compromising the privacy of the individuals by releasing sensitive information such as their medical or financial records, addresses and telephone numbers, preferences of various kinds which the individuals may not want exposed. Merely anonymizing the data is not sufficient – an adversary with access to publicly available auxiliary information can still recover the information about individual, as was the case with the de-anonymization of the Netflix dataset [1].

In this paper, we address the problem of learning a classifier from a multi-party collection of such private data. A set of parties $P_1, P_2, \ldots, P_K$ each possess data $D_1, D_2, \ldots, D_K$. The aim is to learn a classifier from the union of all the data $D_1 \cup D_2 \ldots \cup D_K$. We specifically consider a logistic regression classifier, but as we shall see, the techniques are generally applicable to any classification algorithm. The conditions we impose are that (a) None of the parties are willing to share the data with one another or with any third party (*e.g.* a curator). (b) The computed classifier cannot be reverse engineered to learn about any individual data instance possessed by any contributing party.

The conventional approach to learning functions in this manner is through secure multi-party computation (SMC) [2]. Within SMC individual parties use a combination of cryptographic techniques and oblivious transfer to jointly compute a function of their private data [3, 4, 5]. The techniques typically provide guarantees that none of the parties learn anything about the individual data besides what may be inferred from the final result of the computation. Unfortunately, this does not satisfy condition (b) above. For instance, when the outcome of the computation is a classifier, it does not prevent an adversary from postulating the presence of data instances whose absence might change

the decision boundary of the classifier, and verifying the hypothesis using auxiliary information if any. Moreover, for all but the simplest computational problems, SMC protocols tend to be highly expensive, requiring iterated encryption and decryption and repeated communication of encrypted partial results between participating parties.

An alternative theoretical model for protecting the privacy of individual data instances is *differential privacy* [6]. Within this framework, a stochastic component is added to any computational mechanism, typically by the addition of noise. A mechanism evaluated over a database is said to satisfy differential privacy if the probability of the mechanism producing a particular output is almost the same regardless of the presence or absence of any individual data instance in the database. Differential privacy provides statistical guarantees that the output of the computation does not carry information about individual data instances. On the other hand, in multiparty scenarios where the data used to compute a function are distributed across several parties, it does not provide any mechanism for preserving the privacy of the contributing parties from one another or alternately, from a curator who computes the function from the combined data.

We provide an alternative solution: within our approach the individual parties locally compute an optimal classifier with their data. The individual classifiers are then averaged to obtain the final aggregate classifier. The aggregation is performed through a secure protocol that also adds a stochastic component to the averaged classifier, such that the resulting aggregate classifier is differentially private, *i.e.*, no inference may be made about individual data instances from the classifier. This procedure satisfies both criteria (a) and (b) mentioned above. Furthermore, it is significantly less expensive than any SMC protocol to compute the classifier on the combined data.

We also present theoretical guarantees on the classifier. We provide a fundamental result that the excess risk of an aggregate classifier obtained by averaging classifiers trained on individual subsets, compared to the optimal classifier computed on the combined data in the union of all subsets, is bounded by a quantity that depends on the size of the smallest subset. We prove that the addition of the noise does indeed result in a differentially private classifier. We also provide a bound on the true excess risk of the differentially private averaged classifier compared to the optimal classifier trained on the combined data. Finally, we present experimental evaluation of the proposed technique on a UCI Adult dataset which is a subset of the 1994 census database and empirically show that the differentially private classifier trained using the proposed method provides the performance close to the optimal classifier when the distribution of data across parties is reasonably equitable.

## 2 Differential Privacy

In this paper, we consider the differential privacy model introduced by Dwork [6]. Given any two databases $D$ and $D'$ differing by one element, which we will refer to as *adjacent databases*, a randomized query function $M$ is said to be differentially private if the probability that $M$ produces a response $S$ on $D$ is close to the probability that $M$ produces the same response $S$ on $D'$. As the query output is almost the same in the presence or absence of an individual entry with high probability, nothing can be learned about any individual entry from the output.

**Definition** A randomized function $M$ with a well-defined probability density $P$ satisfies $\epsilon$-differential privacy if, for all adjacent databases $D$ and $D'$ and for any $S \in range(M)$,

$$\left| \log \frac{P\left(M(D) = S\right)}{P\left(M(D') = S\right)} \right| \leq \epsilon. \tag{1}$$

In a classification setting, the training dataset may be thought of as the database and the algorithm learning the classification rule as the query mechanism. A classifier satisfying differential privacy implies that no additional details about the individual training data instances can be obtained with certainty from output of the learning algorithm, beyond the *a priori* background knowledge. Differential privacy provides an *ad omnia* guarantee as opposed to most other models that provide *ad hoc* guarantees against a specific set of attacks and adversarial behaviors. By evaluating the differentially private classifier over a large number of test instances, an adversary cannot learn the exact form of the training data.

## 2.1 Related Work

Dwork *et al.* [7] proposed the *exponential mechanism* for creating functions satisfying differential privacy by adding a perturbation term from the Laplace distribution scaled by the *sensitivity* of the function. Chaudhuri and Monteleoni [8] use the exponential mechanism [7] to create a differentially private logistic regression classifier by perturbing the estimated parameters with multivariate Laplacian noise scaled by the sensitivity of the classifier. They also propose another method to learn classifiers satisfying differential privacy by adding a linear perturbation term to the objective function which is scaled by Laplacian noise. Nissim, *et al.* [9] show we can create a differentially private function by adding noise from Laplace distribution scaled by the *smooth sensitivity* of the function. While this mechanism results in a function with lower error, the smooth sensitivity of a function can be difficult to compute in general. They also propose the *sample and aggregate* framework for replacing the original function with a related function for which the smooth sensitivity can be easily computed. Smith [10] presents a method for differentially private unbiased MLE using this framework.

All the previous methods are inherently designed for the case where a single curator has access to the entire data and is interested in releasing a differentially private function computed over the data. To the best of our knowledge and belief, ours is the first method designed for releasing a differentially private classifier computed over training data owned by different parties who do not wish to disclose the data to each other. Our technique was principally motivated by the sample and aggregate framework, where we considered the samples to be owned by individual parties. Similar to [10], we choose a simple average as the aggregation function and the parties together release the perturbed aggregate classifier which satisfies differential privacy. In the multi-party case, however, adding the perturbation to the classifier is no longer straightforward and it is necessary to provide a secure protocol to do this.

## 3 Multiparty Classification Protocol

The problem we address is as follows: a number of parties $P_1, \ldots, P_K$ possess data sets $D_1, \ldots, D_K$ where $D_i = (\mathbf{x}, \mathbf{y})|_j$ includes a set of instances $\mathbf{x}$ and their binary labels $\mathbf{y}$. We want to train a logistic regression classifier on the combined data such that no party is required to expose any of its data, and the no information about any single data instance can be obtained from the learned classifier. The protocol can be divided into the three following phases:

### 3.1 Training Local Classifiers on Individual Datasets

Each party $P_j$ uses their data set $(\mathbf{x}, \mathbf{y})|_j$ to learn an $\ell_2$ regularized logistic regression classifier with weights $\hat{\mathbf{w}}_j$. This is obtained by minimizing the following objective function

$$\hat{\mathbf{w}}_j = \underset{\mathbf{w}}{\operatorname{argmin}} \; J(\mathbf{w}) = \underset{\mathbf{w}}{\operatorname{argmin}} \; \frac{1}{n_j} \sum_i \log \left( 1 + e^{-y_i \mathbf{w}^T \mathbf{x}_i} \right) + \lambda \mathbf{w}^T \mathbf{w}, \qquad (2)$$

where $\lambda > 0$ is the regularization parameter. Note that no data or information has been shared yet.

### 3.2 Publishing a Differentially Private Aggregate Classifier

The proposed solution, illustrated by Figure 1, proceeds as follows. The parties then collaborate to compute an *aggregate* classifier given by $\hat{\mathbf{w}}^s = \frac{1}{K} \sum_j \hat{\mathbf{w}}_j + \boldsymbol{\eta}$, where $\boldsymbol{\eta}$ is a $d$-dimensional random variable sampled from a Laplace distribution scaled with the parameter $\frac{2}{n_{(1)} \epsilon \lambda}$ and $n_{(1)} = \min_j n_j$. As we shall see later, composing an aggregate classifier in this manner incurs only a well-bounded excess risk over training a classifier directly on the union of all data while enabling the parties to maintain their privacy. We also show in Section 4.1 that the noise term $\boldsymbol{\eta}$ ensures that the classifier $\hat{\mathbf{w}}^s$ satisfies differential privacy, *i.e.*, that individual data instances cannot be discerned from the aggregate classifier. The definition of the noise term $\boldsymbol{\eta}$ above may appear unusual at this stage, but it has an intuitive explanation: A classifier constructed by aggregating locally trained classifiers is limited by the performance of the individual classifier that has the least number of data instances. This will be formalized in Section 4.2. We note that the parties $P_j$ cannot simply take

their individually trained classifiers $\hat{\mathbf{w}}_j$, perturb them with a noise vector and publish the perturbed classifiers, because aggregating such classifiers will not give the correct $\boldsymbol{\eta} \sim \mathrm{Lap}\big(2/(n_{(1)}\epsilon\lambda)\big)$ in general. Since individual parties cannot simply add noise to their classifiers to impose differential privacy, the actual averaging operation must be performed such that the individual parties do not expose their own classifiers or the number of data instances they possess. We therefore use a private multiparty protocol, interacting with an untrusted curator "Charlie" to perform the averaging. The outcome of the protocol is such that each of the parties obtain additive shares of the final classifier $\hat{\mathbf{w}}^s$, such that these shares must be added to obtain $\hat{\mathbf{w}}^s$.

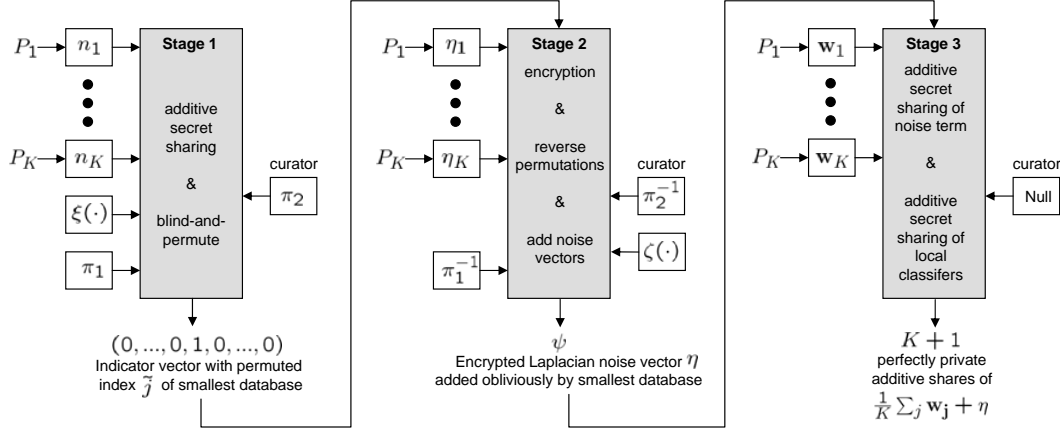

Figure 1: Multiparty protocol to securely compute additive shares of $\hat{\mathbf{w}}^s$.

**Privacy-Preserving Protocol**

We use asymmetric key additively homomorphic encryption [11]. A desirable property of such schemes is that we can perform operations on the ciphertext elements which map into known operations on the same plaintext elements. For an additively homomorphic encryption function $\xi(\cdot)$, $\xi(a)\,\xi(b) = \xi(a+b), \;\; \xi(a)^b = \xi(ab)$. Note that the additively homomorphic scheme employed here is semantically secure, i.e., repeated encryption of the same plaintext will result in different ciphertexts. For the ensuing protocol, encryption keys are considered public and decryption keys are privately owned by the specified parties. Assuming the parties to be honest-but-curious, the steps of the protocol are as follows.

**Stage 1.  Finding the index of the smallest database obfuscated by permutation.**
1. Each party $P_j$ computes $n_j = a_j + b_j$, where $a_j$ and $b_j$ are integers representing additive shares of the database lengths $n_j$ for $j = 1, 2, ..., K$. Denote the $K$-length vectors of additive shares as $\mathbf{a}$ and $\mathbf{b}$ respectively.
2. The parties $P_j$ mutually agree on a permutation $\pi_1$ on the index vector $(1, 2, ..., K)$. This permutation is unknown to Charlie. Then, each party $P_j$ sends its share $a_j$ to party $P_{\pi_1(j)}$, and sends its share $b_j$ to Charlie with the index changed according to the permutation. Thus, after this step, the parties have permuted additive shares given by $\pi_1(\mathbf{a})$ while Charlie has permuted additive shares $\pi_1(\mathbf{b})$.
3. The parties $P_j$ generate a key pair ($pk$,$sk$) where $pk$ is a public key for homomorphic encryption and $sk$ is the secret decryption key known only to the parties but not to Charlie. Denote element-wise encryption of $\mathbf{a}$ by $\xi(\mathbf{a})$. The parties send $\xi(\pi_1(\mathbf{a})) = \pi_1(\xi(\mathbf{a}))$ to Charlie.
4. Charlie generates a random vector $\mathbf{r} = (r_1, r_2, \cdots, r_K)$ where the elements $r_i$ are integers chosen uniformly at random and are equally likely to be positive or negative. Then, he computes $\xi(\pi_1(a_j))\xi(r_j) = \xi(\pi_1(a_j) + r_j)$. In vector notation, he computes $\xi(\pi_1(\mathbf{a}) + \mathbf{r})$. Similarly, by subtracting the same random integers in the same order to his own shares, he obtains $\pi_1(\mathbf{b}) - \mathbf{r}$ where $\pi_1$ was the permutation unknown to him and applied by the parties. Then, Charlie selects a permutation $\pi_2$ at random

and obtains $\pi_2(\xi(\pi_1(\mathbf{a}) + \mathbf{r})) = \xi(\pi_2(\pi_1(\mathbf{a}) + \mathbf{r}))$ and $\pi_2(\pi_1(\mathbf{b}) - \mathbf{r})$. He sends $\xi(\pi_2(\pi_1(\mathbf{a}) + \mathbf{r}))$ to the individual parties in the following order: First element to $P_1$, second element to $P_2$,...,$K^{\text{th}}$ element to $P_K$.

5. Each party decrypts the signal received from Charlie. At this point, the parties $P_1, P_2, ..., P_K$ respectively possess the elements of the vector $\pi_2(\pi_1(\mathbf{a}) + \mathbf{r})$ while Charlie possesses the vector $\pi_2(\pi_1(\mathbf{b}) - \mathbf{r})$. Since $\pi_1$ is unknown to Charlie and $\pi_2$ is unknown to the parties, the indices in both vectors have been complete obfuscated. Note also that, adding the vector collectively owned by the parties and the vector owned by Charlie would give $\pi_2(\pi_1(\mathbf{a}) + \mathbf{r}) + \pi_2(\pi_1(\mathbf{b}) - \mathbf{r}) = \pi_2(\pi_1(\mathbf{a} + \mathbf{b})) = \pi_2(\pi_1(\mathbf{n}))$. This situation in this step is similar to that encountered in the "blind and permute" protocol used for minimum-finding by Du and Atallah [12].

6. Let $\pi_2(\pi_1(\mathbf{a}) + \mathbf{r}) = \tilde{\mathbf{a}}$ and $\pi_2(\pi_1(\mathbf{b}) - \mathbf{r}) = \tilde{\mathbf{b}}$. Then $n_i > n_j \Rightarrow \tilde{a}_i + \tilde{b}_i > \tilde{a}_j + \tilde{b}_j \Rightarrow \tilde{a}_i - \tilde{a}_j > \tilde{b}_j - \tilde{b}_i$. For each $(i,j)$ pair with $i, j \in \{1, 2, ..., K\}$, these comparisons can be solved by any implementation of a secure millionaire protocol [2]. When all the comparisons are done, Charlie finds the index $\tilde{j}$ such that $\tilde{a}_{\tilde{j}} + \tilde{b}_{\tilde{j}} = \min_j n_j$. The *true* index corresponding to the smallest database has already been obfuscated by the steps of the protocol. Charlie holds only an additive share of $\min_j n_j$ and thus cannot know the true length of the smallest database.

**Stage 2. Obliviously obtaining encrypted noise vector from the smallest database.**

1. Charlie constructs an $K$ indicator vector $\mathbf{u}$ such that $u_{\tilde{j}} = 1$ and all other elements are 0. He then obtains the permuted vector $\pi_2^{-1}(\mathbf{u})$ where $\pi_2^{-1}$ inverts $\pi_2$. He generates a key-pair $(pk',sk')$ for additive homomorphic function $\zeta(\cdot)$ where only the encryption key $pk'$ is publicly available to the parties $P_j$. Charlie then transmits $\zeta(\pi_2^{-1}(\mathbf{u})) = \pi_2^{-1}(\zeta(\mathbf{u}))$ to the parties $P_j$.

2. The parties mutually obtain a permuted vector $\pi_1^{-1}(\pi_2^{-1}(\zeta(\mathbf{u}))) = \zeta(\mathbf{v})$ where $\pi_1^{-1}$ inverts the permutation $\pi_1$ originally applied by the parties $P_j$ in Stage I. Now that both permutations have been removed, the index of the non-zero element in the indicator vector $\mathbf{v}$ corresponds to the true index of the smallest database. However, since the parties $P_j$ cannot decrypt $\zeta(\cdot)$, they cannot find out this index.

3. For $j = 1, \ldots, K$, party $P_j$ generates $\boldsymbol{\eta}_j$, a $d$-dimensional noise vector sampled from a Laplace distribution with parameter $\frac{2}{n_j \epsilon \lambda}$. Then, it obtains a $d$-dimensional vector $\boldsymbol{\psi}_j$ where for $i = 1, \ldots, d$, $\psi_j(i) = \zeta(v(j))^{\eta_j(i)} = \zeta(v(j)\,\eta_j(i))$.

4. All parties $P_j$ now compute a $d$-dimensional noise vector $\boldsymbol{\psi}$ such that, for $i = 1, \ldots, d$,
$$\psi(i) = \prod_j \psi_j(i) = \prod_j \zeta(v(j)\eta_j(i)) = \zeta\left(\sum_j v(j)\eta_j(i)\right).$$
The reader will notice that, by construction, the above equation selects only the Laplace noise terms for the smallest database, while rejecting the noise terms for all other databases. This is because $\mathbf{v}$ has an element with value 1 at the index corresponding to the smallest database and has zeroes everywhere else. Thus, the decryption of $\boldsymbol{\psi}$ is equal to $\boldsymbol{\eta}$ which was the desired perturbation term defined at the beginning of this section.

**Stage 3. Generating secret additive shares of $\hat{\mathbf{w}}^s$.**

1. One of the parties, say $P_1$, generates a $d$-dimensional random integer noise vector $\mathbf{s}$, and transmits $\psi(i)\zeta(s(i))$ for all $i = 1, \ldots, d$ to Charlie. Using $\mathbf{s}$ effectively prevents Charlie from discovering $\boldsymbol{\eta}$, and therefore still ensures that no information is leaked about the database owners $P_j$. $P_1$ computes $\mathbf{w}_1 - K\mathbf{s}$.

2. Charlie decrypts $\psi(i)\zeta(s(i))$ to obtain $\eta(i) + s(i)$ for $i = 1, \ldots, d$. At this stage, the parties and Charlie have the following $d$-dimensional vectors: Charlie has $K(\boldsymbol{\eta} + \mathbf{s})$, $P_1$ has $\hat{\mathbf{w}}_1 - K\mathbf{s}$, and all other parties $P_j$, $j = 2, \ldots, K$ have $\hat{\mathbf{w}}_j$. None of the $K + 1$ participants can share this data for fear of compromising differential privacy.

3. Finally, Charlie and the $K$ database-owning parties run a simple secure function evaluation protocol [13], at the end of which each of the $K + 1$ participants obtains an additive share of $K\hat{\mathbf{w}}^s$. This protocol is provably private against honest but curious participants when there are no collisions. The resulting shares are published.

The above protocol ensures the following (a) None of the $K+1$ participants, or users of the perturbed aggregate classifier can find out the size of any database, and therefore none of the parties knows who contributed $\boldsymbol{\eta}$ (b) Neither Charlie nor any of the parties $P_j$ can individually remove the noise $\boldsymbol{\eta}$ after the additive shares are published. This last property is important because if anyone knowingly could remove the noise term, then the resulting classifier no longer provides differential privacy.

### 3.3 Testing Phase

A test participant Dave having a test data instance $\mathbf{x}' \in \mathbb{R}^d$ is interested in applying the trained classifier adds the published shares and divides by $K$ to get the differentially private classifier $\hat{\boldsymbol{w}}^s$. He can then compute the sigmoid function $t = \frac{1}{1+e^{-\hat{\boldsymbol{w}}^{sT}\mathbf{x}_i}}$ and decide to classify $\mathbf{x}'$ with label $-1$ if $t \leq \frac{1}{2}$ and with label $1$ if $t > \frac{1}{2}$.

## 4 Theoretical Analysis

### 4.1 Proof of Differential Privacy

We show that the perturbed aggregate classifier satisfies differential privacy. We use the following bound on the sensitivity of the regularized regression classifier as proved in Corollary 2 in [8] restated in the appendix as Theorem 6.1.

**Theorem 4.1.** *The classifier $\hat{\boldsymbol{w}}^s$ preserves $\epsilon$-differential privacy. For any two adjacent datasets $D$ and $D'$,*

$$\left| \log \frac{P(\hat{\boldsymbol{w}}^s|D)}{P(\hat{\boldsymbol{w}}^s|D')} \right| \leq \epsilon.$$

*Proof.* Consider the case where one instance of the training dataset $D$ is changed to result in an adjacent dataset $D'$. This would imply a change in one element in the training dataset of one party and thereby a change in the corresponding learned vector $\hat{\mathbf{w}}_j^s$. Assuming that the change is in the dataset of the party $P_j$, the change in the learned vector is only going to be in $\hat{\mathbf{w}}_j$; let denote the new classifier by $\hat{\mathbf{w}}_j'$. In Theorem 6.1, we bound the sensitivity of $\hat{\mathbf{w}}_j$ as $\|\hat{\mathbf{w}}_j - \hat{\mathbf{w}}_j'\|_1 \leq \frac{2}{n_j\epsilon\lambda}$. Following an argument similar to [7], considering that we learn the same vector $\hat{\boldsymbol{w}}^s$ using either the training datasets $D$ and $D'$, we have

$$\frac{P(\hat{\mathbf{w}}^s|D)}{P(\hat{\mathbf{w}}^s|D')} = \frac{P(\hat{\mathbf{w}}_j + \boldsymbol{\eta}|D)}{P(\hat{\mathbf{w}}_j' + \boldsymbol{\eta}|D')} = \frac{\exp\left[\frac{n_{(1)}\epsilon\lambda}{2}\|\hat{\mathbf{w}}_j\|_1\right]}{\exp\left[\frac{n_{(1)}\epsilon\lambda}{2}\|\hat{\mathbf{w}}_j'\|_1\right]} \leq \exp\left[\frac{n_{(1)}\epsilon\lambda}{2}\|\hat{\mathbf{w}}_j - \hat{\mathbf{w}}_j'\|_1\right]$$

$$\leq \exp\left[\frac{n_{(1)}\epsilon\lambda}{2}\frac{2}{n_j\lambda}\right] \leq \exp\left[\frac{n_{(1)}}{n_j}\epsilon\right] \leq \exp(\epsilon),$$

by the definition of function sensitivity. Similarly, we can lower bound the the ratio by $\exp(-\epsilon)$. □

### 4.2 Analysis of Excess Error

In the following discussion, we consider how much excess error is introduced when using a perturbed aggregate classifier $\hat{\boldsymbol{w}}^s$ satisfying differential privacy as opposed to the unperturbed classifier $\mathbf{w}^*$ trained on the entire training data while ignoring the privacy constraints as well as the unperturbed aggregate classifier $\hat{\mathbf{w}}$.

We first establish a bound on the $\ell_2$ norm of the difference between the aggregate classifier $\hat{\mathbf{w}}$ and the classifier $\mathbf{w}^*$ trained over the entire training data. To prove the bound we apply Lemma 1 from [8] restated as Lemma 6.2 in the appendix. Please refer to the appendix for the proof of the following theorem.

**Theorem 4.2.** *Given the aggregate classifier $\hat{\boldsymbol{w}}$, the classifier $\boldsymbol{w}^*$ trained over the entire training data and $n_{(1)}$ is the size of the smallest training dataset,*

$$\|\hat{\boldsymbol{w}} - \boldsymbol{w}^*\|_2 \leq \frac{K-1}{n_{(1)}\lambda}.$$

The bound is inversely proportional to the number of instances in the smallest dataset. This indicates that when the datasets are of disparate sizes, $\hat{\mathbf{w}}$ will be a lot different from $\mathbf{w}^*$. The largest possible value for $n_{(1)}$ is $\frac{n}{K}$ in which case all parties having an equal amount of training data and $\hat{\mathbf{w}}$ will be closest to $\mathbf{w}^*$. In the one party case for $K = 1$, the bound indicates that norm of the difference would be upper bounded by zero, which is a valid sanity check as the aggregate classifier $\hat{\mathbf{w}}$ is the same as $\mathbf{w}^*$.

We use this result to establish a bound on the empirical risk of the perturbed aggregate classifier $\hat{\mathbf{w}}^s = \hat{\mathbf{w}} + \boldsymbol{\eta}$ over the empirical risk of the unperturbed classifier $\mathbf{w}^*$ in the following theorem. Please refer to the appendix for the proof.

**Theorem 4.3.** *If all data instances $\mathbf{x}_i$ lie in a unit ball, with probability at least $1 - \delta$, the empirical regularized excess risk of the perturbed aggregate classifier $\hat{\mathbf{w}}^s$ over the classifier $\mathbf{w}^*$ trained over entire training data is*

$$J(\hat{\boldsymbol{w}}^s) \leq J(\boldsymbol{w}^*) + \frac{(K-1)^2(\lambda+1)}{2n_{(1)}^2\lambda^2} + \frac{2d^2(\lambda+1)}{n_{(1)}^2\epsilon^2\lambda^2}\log^2\left(\frac{d}{\delta}\right) + \frac{2d(K-1)(\lambda+1)}{n_{(1)}^2\epsilon\lambda^2}\log\left(\frac{d}{\delta}\right).$$

The bound suggests an error because of two factors: aggregation and perturbation. The bound increases for smaller values of $\epsilon$ implying a tighter definition of differential privacy, indicating a clear trade-off between privacy and utility. The bound is also inversely proportional to $n_{(1)}^2$ implying an increase in excess risk when the parties have training datasets of disparate sizes.

In the limiting case $\epsilon \rightarrow \infty$, we are adding a perturbation term $\boldsymbol{\eta}$ sampled from a Laplacian distribution of infinitesimally small variance resulting in the perturbed classifier being almost as same as using the unperturbed aggregate classifier $\hat{\mathbf{w}}$ satisfying a very loose definition of differential privacy. With such a value of $\epsilon$, our bound becomes

$$J(\hat{\mathbf{w}}) \leq J(\mathbf{w}^*) + \frac{(K-1)^2(\lambda+1)}{2n_{(1)}^2\lambda^2}. \tag{3}$$

Similar to the analysis of Theorem 4.2, the excess error in using an aggregate classifier is inversely proportional to the size of the smallest dataset $n_{(1)}$ and in the one party case $K = 1$, the bound becomes zero as the aggregate classifier $\hat{\mathbf{w}}$ is the same as $\mathbf{w}^*$. Also, for a small value of $\epsilon$ in the one party case $K = 1$ and $n_{(1)} = n$, our bound reduces to that in Lemma 3 of [8],

$$J(\hat{\mathbf{w}}^s) \leq J(\mathbf{w}^*) + \frac{2d^2(\lambda+1)}{n^2\epsilon^2\lambda^2}\log^2\left(\frac{d}{\delta}\right). \tag{4}$$

While the previous theorem gives us a bound on the empirical excess risk over a given training dataset, it is important to consider a bound on the true excess risk of $\hat{\mathbf{w}}^s$ over $\mathbf{w}^*$. Let us denote the true risk of the classifier $\hat{\mathbf{w}}^s$ by $\tilde{J}(\hat{\mathbf{w}}^s) = \mathbb{E}[J(\hat{\mathbf{w}}^s)]$ and similarly, the true risk of the classifier $\mathbf{w}^*$ by $\tilde{J}(\mathbf{w}^*) = \mathbb{E}[J(\mathbf{w}^*)]$. In the following theorem, we apply the result from [14] which uses the bound on the empirical excess risk to form a bound on the true excess risk. Please refer to the appendix for the proof.

**Theorem 4.4.** *If all training data instances $\mathbf{x}_i$ lie in a unit ball, with probability at least $1 - \delta$, the true excess risk of the perturbed aggregate classifier $\hat{\mathbf{w}}^s$ over the classifier $\mathbf{w}^*$ trained over entire training data is*

$$\tilde{J}(\hat{\boldsymbol{w}}^s) \leq \tilde{J}(\boldsymbol{w}^*) + \frac{2(K-1)^2(\lambda+1)}{2n_{(1)}^2\lambda^2} + \frac{4d^2(\lambda+1)}{n_{(1)}^2\epsilon^2\lambda^2}\log^2\left(\frac{d}{\delta}\right)$$
$$+ \frac{4d(K-1)(\lambda+1)}{n_{(1)}^2\epsilon\lambda^2}\log\left(\frac{d}{\delta}\right) + \frac{16}{\lambda n}\left[32 + \log\left(\frac{1}{\delta}\right)\right].$$

## 5   Experiments

We perform an empirical evaluation of the proposed differentially private classifier to obtain a characterization of the increase in the error due to perturbation. We use the Adult dataset from the UCI machine learning repository [15] consisting of personal information records extracted from

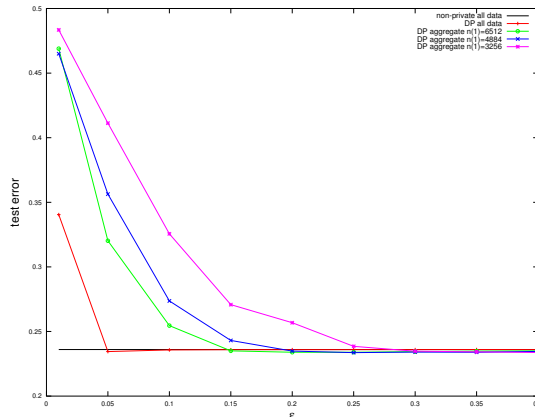

Figure 2: Classifier performance evaluated for $\mathbf{w}^*$, $\mathbf{w}^* + \boldsymbol{\eta}$, and $\hat{\mathbf{w}}^s$ for different data splits vs. $\epsilon$

the census database and the task is to predict whether a given person has an annual income over $50,000. The choice of the dataset is motivated as a realistic example for application of data privacy techniques. The original Adult data set has six continuous and eight categorical features. We use pre-processing similar to [16], the continuous features are discretized into quintiles, and each quintile is represented by a binary feature. Each categorical feature is converted to as many binary features as its cardinality. The dataset contains 32,561 training and 16,281 test instances each with 123 features.[1] In Figure 2, we compare the test error of perturbed aggregate classifiers trained over data from five parties for different values of $\epsilon$. We consider three situations: all parties with equal datasets containing 6512 instances (even split, $n_{(1)} = 20\%$ of $n$), parties with datasets containing 4884, 6512, 6512, 6512, 8141 instances ($n_{(1)} = 15\%$ of $n$), and parties with datasets containing 3256, 6512, 6512, 6512, 9769 instances ($n_{(1)} = 10\%$ of $n$). We also compare with the error of the classifier trained using combined training data and its perturbed version satisfying differential privacy. We chose the value of the regularization parameter $\lambda = 1$ and the results displayed are averaged over 200 executions.

The perturbed aggregate classifier which is trained using maximum $n_{(1)} = 6512$ does consistently better than for lower values of $n_{(1)}$ which is same as our theory suggested. Also, the test error for all perturbed aggregate classifiers drops with $\epsilon$, but comparatively faster for even split and converges to the test error of the classifier trained over the combined data. As expected, the differentially private classifier trained over the entire training data does much better than the perturbed aggregate classifiers with an error equal to the unperturbed classifier except for small values of $\epsilon$. The lower error of this classifier is at the cost of the loss in privacy of the parties as they would need to share the data in order to train the classifier over combined data.

## 6  Conclusion

We proposed a method for composing an aggregate classifier satisfying $\epsilon$-differential privacy from classifiers locally trained by multiple untrusting parties. The upper bound on the excess risk of the perturbed aggregate classifer as compared to the optimal classifier trained over the complete data without privacy constraints is inversely proportional to the privacy parameter $\epsilon$, suggesting an inherent tradeoff between privacy and utility. The bound is also inversely proportional to the size of the smallest training dataset, implying the best performance when the datasets are of equal sizes. Experimental results on the UCI Adult data also show the behavior suggested by the bound and we observe that the proposed method provides classification performance close to the optimal non-private classifier for appropriate values of $\epsilon$. In future work, we seek to generalize the theoretical analysis of the perturbed aggregate classifier to the setting in which each party has data generated from a different distribution.

## Footnotes

[1]The dataset can be download from http://www.csie.ntu.edu.tw/ cjlin/libsvmtools/datasets/binary.html#a9a

# References

[1] Arvind Narayanan and Vitaly Shmatikov. De-anonymizing social networks. In *IEEE Symposium on Security and Privacy*, pages 173–187, 2009.

[2] Andrew Yao. Protocols for secure computations (extended abstract). In *IEEE Symposium on Foundations of Computer Science*, 1982.

[3] Jaideep Vaidya, Chris Clifton, Murat Kantarcioglu, and A. Scott Patterson. Privacy-preserving decision trees over vertically partitioned data. *TKDD*, 2(3), 2008.

[4] Jaideep Vaidya, Murat Kantarcioglu, and Chris Clifton. Privacy-preserving naive bayes classification. *VLDB J*, 17(4):879–898, 2008.

[5] Jaideep Vaidya, Hwanjo Yu, and Xiaoqian Jiang. Privacy-preserving svm classification. *Knowledge and Information Systems*, 14(2):161–178, 2008.

[6] Cynthia Dwork. Differential privacy. In *International Colloquium on Automata, Languages and Programming*, 2006.

[7] Cynthia Dwork, Frank McSherry, Kobbi Nissim, and Adam Smith. Calibrating noise to sensitivity in private data analysis. In *Theory of Cryptography Conference*, pages 265–284, 2006.

[8] Kamalika Chaudhuri and Claire Monteleoni. Privacy-preserving logistic regression. In *Neural Information Processing Systems*, pages 289–296, 2008.

[9] Kobbi Nissim, Sofya Raskhodnikova, and Adam Smith. Smooth sensitivity and sampling in private data analysis. In *ACM Symposium on Theory of Computing*, pages 75–84, 2007.

[10] Adam Smith. Efficient, differentially private point estimators. *arXiv:0809.4794v1 [cs.CR]*, 2008.

[11] Pascal Paillier. Public-key cryptosystems based on composite degree residuosity classes. In *EUROCRYPT*, 1999.

[12] Mikhail Atallah and Jiangtao Li. Secure outsourcing of sequence comparisons. *International Journal of Information Security*, 4(4):277–287, 2005.

[13] Michael Ben-Or, Shari Goldwasser, and Avi Widgerson. Completeness theorems for non-cryptographic fault-tolerant distributed computation. In *Proceedings of the ACM Symposium on the Theory of Computing*, pages 1–10, 1988.

[14] Karthik Sridharan, Shai Shalev-Shwartz, and Nathan Srebro. Fast rates for regularized objectives. In *Neural Information Processing Systems*, pages 1545–1552, 2008.

[15] A. Frank and A. Asuncion. UCI machine learning repository, 2010.

[16] John Platt. Fast training of support vector machines using sequential minimal optimization. In *Advances in Kernel Methods — Support Vector Learning*, pages 185–208, 1999.

